# TinyTTA: Efficient Test-Time Adaptation via Early-Exit Ensembles on Edge Devices

**Hong Jia**[1,3]    **Young D. Kwon**[1,4]    **Alessio Orsino**[1,2]    **Ting Dang**[3]
**Domenico Talia**[2]    **Cecilia Mascolo**[1]
[1]University of Cambridge    [2]University of Calabria    [3]University of Melbourne
[4]Samsung AI Center, Cambridge
{hj359,ydk21,cm542}@cam.ac.uk
{aorsino,talia}@dimes.unical.it
{hong.jia,ting.dang}@unimelb.edu.au

## Abstract

The increased adoption of Internet of Things (IoT) devices has led to the generation of large data streams with applications in healthcare, sustainability, and robotics. In some cases, deep neural networks have been deployed directly on these resource-constrained units to limit communication overhead, increase efficiency and privacy, and enable real-time applications. However, a common challenge in this setting is the continuous adaptation of models necessary to accommodate changing environments, i.e., data distribution shifts. Test-time adaptation (TTA) has emerged as one potential solution, but its validity has yet to be explored in resource-constrained hardware settings, such as those involving microcontroller units (MCUs). TTA on constrained devices generally suffers from $i$) memory overhead due to the full backpropagation of a large pre-trained network, $ii$) lack of support for normalization layers on MCUs, and $iii$) either memory exhaustion with large batch sizes required for updating or poor performance with small batch sizes. In this paper, we propose TinyTTA, to enable, for the first time, efficient TTA on constrained devices with limited memory. To address the limited memory constraints, we introduce a novel self-ensemble and batch-agnostic early-exit strategy for TTA, which enables continuous adaptation with small batch sizes for reduced memory usage, handles distribution shifts, and improves latency efficiency. Moreover, we develop the TinyTTA Engine, *a first-of-its-kind* MCU library that enables on-device TTA. We validate TinyTTA on a Raspberry Pi Zero 2W and an STM32H747 MCU. Experimental results demonstrate that TinyTTA improves TTA accuracy by up to 57.6%, reduces memory usage by up to six times, and achieves faster and more energy-efficient TTA. Notably, TinyTTA *is the only framework* able to run TTA on MCU STM32H747 with a 512 KB memory constraint while maintaining high performance.

## 1 Introduction

Deploying deep neural networks on IoT devices, such as microcontroller units (MCUs), holds significant potential in many applications requiring real-time data analysis and low-latency responses, such as real-time human health monitoring [1] and robotics [2]. However, a practical challenge in deploying these models on MCUs in real-world scenarios is their adaptation capability. The data encountered in real settings often exhibit distribution shifts due to unforeseen noise and environment. For instance, sensor data can be affected by natural fluctuations and distortions, including weather changes (e.g., fog and snow) and sensor-related issues (e.g., defocus blur) [3, 4]. Existing mitigation strategies, such as fine-tuning [5], typically necessitate updating the entire model, which may be

impractical for edge devices due to constraints on memory, energy, and computational resources. Consequently, developing effective adaptation techniques for edge devices to address such distribution shifts while maintaining efficiency is a critical and pressing need.

Test-time adaptation (TTA) emerges as an effective strategy to mitigate generalization issues [6]. By adapting a pre-trained model to *unlabeled* test samples, TTA aims to align the model with the current data distribution through *entropy minimization*. This technique reduces the model's prediction entropy without requiring extensive retraining [6]. However, conventional TTA approaches face several limitations that prevent them from being applicable to edge devices. Firstly, TTA typically requires updates to either the entire network or several layers. Even when updating only a few layers, it still incurs substantial memory overhead due to the need for backpropagation. This process requires storing activations for a large pre-trained model, therefore imposing substantial memory requirements. Secondly, few works have considered mixed distribution shifts and have uniformly updated networks for samples exhibiting varying distribution shifts. This approach not only results in varying accuracy for different levels of distribution shifts but also increases computational overhead, as samples with minor shifts may not, in theory, necessitate updating the entire network. Thirdly, the most effective TTA strategies often involve adapting the normalization layers [3, 7, 8]. However, in practice, these layers are usually fused with convolutional layers when developed for MCUs to reduce computational and memory demands [9–11]. This poses practical challenges as currently there is no available library support for normalization layers in such hardware environments. Lastly, the effectiveness of normalization-based TTA typically relies on large batch sizes, such as 64 instances per update [3], which significantly increases computational overhead. Instead, constrained hardware such as MCUs typically only allow a single batch of data due to limited memory resources. While smaller batch sizes could be more practical for adaptation in MCUs [5], existing approaches suffer from significant performance degradation under these conditions. *As a result, there is still a lack of efficient TTA frameworks applicable to resource-constrained edge devices. This challenge is even more pronounced for extremely constrained devices, such as MCUs, which play a pivotal role in many applications.*

In this paper, *we present a novel TTA framework named TinyTTA, which, for the first time, addresses the challenges posed by the efficiency requirements of constrained devices to enable TTA.* We show how our design choices allow the framework to apply even to the extreme edge, i.e., MCUs. To address the first challenge, we propose an approach based on self-ensemble and early exits. With self-ensembles, we innovatively group subsequent layers of the pre-trained network into submodules, approximating each submodule to emulate the full network's capability. The early-exit strategy, combined with these submodules, allows instances to exit from different submodules rather than passing through the entire pre-trained network at test time. For example, samples that are easily predicted with high confidence, which is calculated from the model output, can exit from earlier submodules, alleviating the overhead associated with traversing subsequent layers and reducing the need for further propagation through the entire network. This design also accommodates mixed distribution shifts, addressing the second challenge. Samples with minor distribution shifts can be easily predicted, making a shallow submodule sufficient for reliable prediction, thus facilitating an early exit. Conversely, samples with significant distribution shifts may necessitate traversing the entire network to achieve reliable adaptation. To address the third and fourth challenges, we propose a weight standardization (WS)-based [12] adaptation for TTA. This method facilitates batch-agnostic TTA by simulating the effects of normalization layers while alleviating the implementation complexity associated with those layers. Additionally, we construct an operator library, named *TinyTTA Engine*, to enable practical deployment on MCUs.

We conducted extensive experiments to assess TinyTTA performance using four benchmark corruption datasets on two edge devices: the Microprocessor (MPU) Raspberry Pi Zero 2W with 512 MB DRAM and STM32H747 MCU with 512 KB of SRAM. Results show that TinyTTA improves TTA accuracy by up to 57.6%, reduces memory usage by up to six times, achieves faster latency, and is more energy-efficient than conventional baseline methods on MPUs. Notably, TinyTTA is the only framework able to run on the MCU STM32H747 with a 512 KB memory constraint while maintaining high performance, opening the door, for the first time, to performing TTA on resource-limited devices.

## 2 Related Work

This section reviews the key methods for performing TTA, emphasizing their main limitations. Subsequently, we examine the literature on early exit mechanisms.

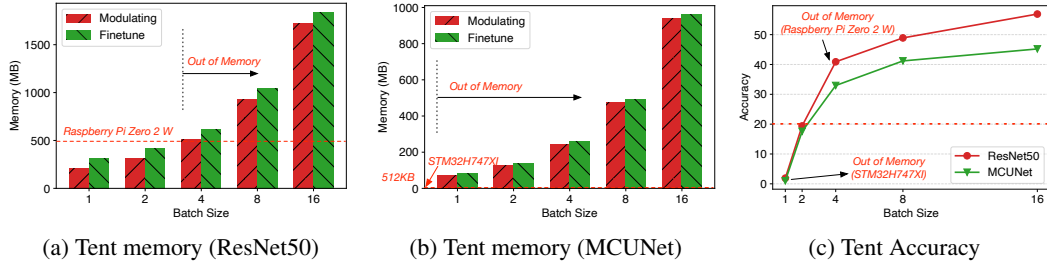

| (a) Tent memory (ResNet50) | (b) Tent memory (MCUNet) | (c) Tent Accuracy |

Figure 1: Motivation study. (a)-(b) Both modulating and fine-tuning TTA have similar memory usage and remain memory-intensive, leading to out-of-memory issues during deployment on MPUs and MCUs. (c) TTA accuracy is highly reliant on batch sizes.

**Memory-intensive TTA.** Existing TTA work can be categorized into fine-tuning TTA and modulating TTA. In particular, fine-tuning TTA methods, such as CoTTA [13] and Test-Time Training (TTT) [14], updates the entire model architecture, leading to intensive memory usage. Modulating TTA methods, instead, adjust normalization layers while keeping other components frozen, as seen in TENT [6], TTN [15], NOTE [16], and SoTTA [17]. However, as shown in Figure 1a and 1b, modulating TTA still exhibits memory usage similar to fine-tuning TTA, since the model still needs to backpropagate through the full model structure.

**Memory-efficient TTA.** To address the memory inefficiency of previous TTA methods, various memory-efficient TTA techniques have been proposed, primarily targeting GPUs. These include EATA [7], which filters out high-entropy data to minimize optimization cache, and MECTA [18], which stops backpropagation caching. However, their analyses remain theoretical and impractical, and memory usage has been shown to be similar to that of modulating TTA [13]. Even the most efficient TTA method, EcoTTA [13], which achieves memory efficiency on GPUs by distilling the pre-trained model via shallower models, still relies on updating normalization layers, which are not supported in MCUs.

**TTA under diverse batch sizes.** As shown in Figure 1c, the success of most TTA methods relies on large batch sizes, due to more reliable statistics provided for normalization adaptation. Several methods target batch size 1 for more efficient adaptation, by using different normalization techniques like group norm (GN) [8], layer norm (LN) [3], and adaptive norm (AdaptBN) [13]. However, all these methods still exhibit poor TTA performance and require updating many normalization layers, which is computationally expensive and impractical for MCUs.

**TTA under changing distribution shifts.** Existing TTA methods often overlook the dynamic nature of distribution shifts and apply the same adaptation principle across all samples. However, real-world data on MCUs, such as sensor readings, exhibit highly diverse distribution shifts over time [19]. To date, only a few studies have considered the presence of different levels of distribution shifts. For example, NOTE [16] and DELTA [20] employ modified normalization layers, while SAR [3] proposes sharpness-aware TTA. However, these methods still rely on normalization layers and thus are not applicable when the batch size is one for MCU deployment.

**Early exit.** Early exit strategies have been explored to reduce computational costs in various machine learning architectures, particularly for real-time and resource-constrained applications. Notably, BranchyNet [21] introduced early exit mechanisms for deep neural networks, enabling faster inference by adding exit points and allowing models to make predictions at intermediate layers instead of traversing the full model for every input. Similarly, Shallow-Deep Networks [22] aim to reduce inference costs by dynamically selecting the exit point based on input complexity. For these reasons, early exits in the TTA setting can offer significant computational benefits. However, most early exit methods have not been designed to address distribution shifts or batch size constraints typical of MCUs.

## 3 Efficient TTA via Early-exit Ensembles and TinyTTA Engine

TinyTTA is designed for continuous adaptation to various types of distribution shifts in data during inference on resource-constrained edge devices. This novel framework, as shown in Figure 2, aims

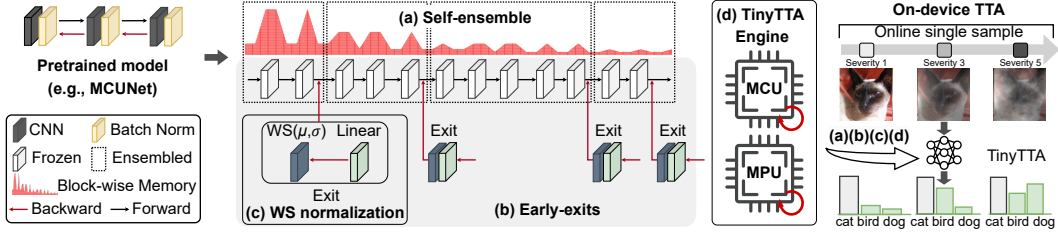

Figure 2: TinyTTA framework overview: TinyTTA (a) begins by analyzing and partitioning a pretrained network into submodules based on memory usage similarity, named as self-ensemble network, (b) introduces early-exits for sample inference through specific submodules based on predicted confidence levels, (c) integrates WS normalization into each submodule to achieve batch-agnostic normalization, and (d) ultimately compiles and deploys the model on MCUs and MPUs for on-device TTA through the TinyTTA Engine.

to improve system memory efficiency during TTA, adaptability of the pre-trained model to diverse distribution shifts, and deployment feasibility on edge devices. TinyTTA comprises four modules: $i$) a self-ensemble network that partitions the network into submodules, each approximating the capability of the full model; $ii$) early-exits, which allow samples to be inferred through specific submodules based on predicted confidence levels, optimizing different submodules for memory and computational efficiency while managing varying levels of distribution shifts; $iii$) WS normalization, integrated into each submodule to achieve batch-agnostic normalization; and $iv$) TinyTTA Engine, an MCU library for on-device TTA.

## 3.1 Self-ensemble network

For edge devices such as MCUs, it has been observed that sharing operational layers and weights, for instance through TensorFlow Lite Micro (TFLM) [11], can significantly reduce on-chip memory usage. Instead of utilizing a pre-trained network in its entirety, partitioning it into submodules can offer substantial benefits in terms of memory usage. Consequently, we propose self-ensembling pre-trained models by dividing layers into submodules, grouping similar subsequent layers, and approximating each submodule with the full model's capabilities. A critical decision in this process is determining the number of partitions required and how to effectively divide the network into submodules. Evidence suggests that $i$) adjacent layers in neural networks tend to exhibit high feature similarities and $ii$) these layers often have comparable memory consumption [5]. This provides a basis for grouping layers with similar memory consumption together as submodules.

We initially conducted an analysis of memory usage during TTA across the layers of a pre-trained model, ResNet50. This analysis, as illustrated in Figure 3, compares memory usage for both activations and weights between fine-tuning (top figure) and modulation (bottom figure) TTA, aiming to identify layers with similar memory usage. We observed the following: $i$) memory usage is primarily driven by activations, which store the outputs at each layer for computing gradients during backpropagation, while the memory consumed by weights is negligible, therefore we focus on activation memory for self-ensembling; $ii$) in both methods, activations share similar memory consumption and are predominantly concentrated in the initial layers, which generally capture crucial information from the input; $iii$) certain groups of adjacent layers, specifically layers 1-15, 16-28, 29-44, and 45-52, show similar sizes of activations. Based on this analysis, we group layers with similar memory usage into submodules (i.e., layers 1-15 for submodule 1, 16-28 for submodule 2, 29-44 for submodule 3, and 45-52 for submodule 4) for subsequent early exits and only update the heads at early exits, freezing the submodules to improve memory usage. Indeed, this process does not require backpropagation to the submodules, thus eliminating the need for activation memory. It is worth highlighting that this analysis shows similar patterns for different model architectures, as discussed in Appendix D.

## 3.2 Early-exits

To further optimize memory usage and manage changing distribution shifts, we introduced early-exit inference combined with self-ensembling submodules. For a given pre-trained model, we sequentially pass inputs through cascaded submodules and infer a confidence level, calculated from the model's

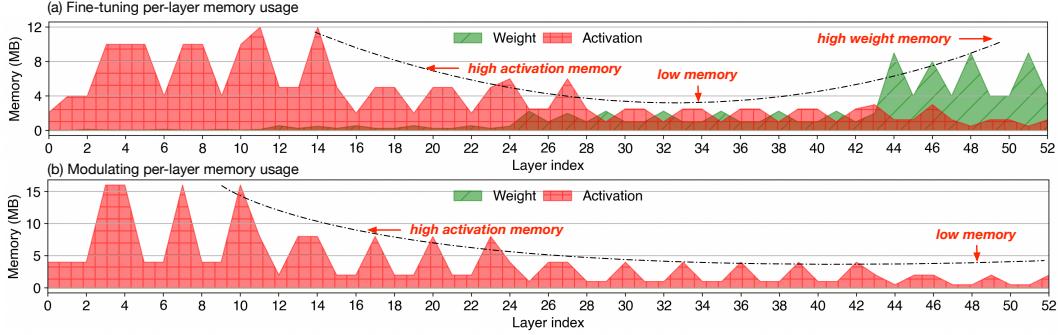

Figure 3: Memory usage of different layers during test-time adaptation. Note that the weight memory for modulation (bottom Figure) is < 10 KB, which is negligible. However, both methods show significant activation memory usage. All experiments were conducted using ResNet50 on the CIFAR-10 dataset with batch size 1.

prediction, for the predictions at each stage. If the confidence level is high and exceeds a predefined threshold, which is a hyper-parameter during TTA (discussed in Appendix E), a good prediction is achieved, allowing the process to exit at that submodule without passing through the remaining ones. Model updates only occur for the preceding submodules. Conversely, if the confidence level remains low, indicating difficulty in prediction or a potential high distribution shift, the instance continues to pass through the subsequent submodules for further inference until it exits from an appropriate submodule with high confidence. This approach saves memory for easy samples that do not require extensive processing, while effectively handling varying distribution shifts. Different samples are processed differently, and updates are optimized according to the varying levels of distribution shifts in the samples. Our work is the first to introduce early exits for on-device TTA in edge devices, enhancing both inference efficiency and model accuracy with data distribution shifts.

Specifically, assume $K$ submodules are derived, denoted by $\Theta = \{\boldsymbol{\theta}^k\}_{k=1}^K$. Each of the submodules is further associated with exits/classifiers, denoted as $\Phi = \{\phi^k\}_{k=1}^K$, using a linear layer. Given the $i^{\text{th}}$ instance, the output for each submodule is first computed as:

$$\boldsymbol{p}_i^k = \frac{\exp\left(\boldsymbol{z}_i^k\right)}{\sum_{j=1}^C \exp\left(\boldsymbol{z}_j^k\right)} \tag{1}$$

where $\boldsymbol{z}_i^k \in \mathbb{R}^C$ denotes the latent representation learned from the $k^{\text{th}}$ submodule, and $\boldsymbol{p}_i^k$ represents the predicted probability for $C$ classes for the classifier of the $k^{\text{th}}$ submodule after the softmax layer. Given the predicted probability $\boldsymbol{p}_i^k$, we can estimate the confidence level using the entropy as follows:

$$H(\boldsymbol{x}_i) = -\sum_{j=1}^C p_j^k \log p_j^k \tag{2}$$

A low entropy indicates low uncertainty in the prediction, thus a high confidence level. If $H(\boldsymbol{x}_i)$ is smaller than the threshold $\gamma^k$ for each submodule $k$, this facilitates an early exit of the instance and also the adaptation of the submodules preceding this $k^{\text{th}}$ submodule. Otherwise, we further process the instance with the subsequent submodules.

### 3.3 WS normalization

To entirely omit the usage of normalization layers and enable accurate and memory-efficient TTA on MCUs, we introduce Weight Standardization (WS) as a replacement for traditional normalization layers [3, 7, 8] *for the first time in TTA*. Particularly, we propose the inclusion of a WS layer preceding the linear layer attached to each submodule. Therefore, the exits/classifiers of each submodule $\phi^k$ consist of a WS layer and a linear layer for $C$ class classification. Our method adapts only $\phi^k$, i.e., the WS statistics and one linear layer during the testing phase, while keeping the other components $\theta^k$ frozen. This avoids model collapse with small batch sizes and minimizes memory usage.

Unlike traditional normalization methods like Batch Normalization (BN) and Group Normalization (GN), which primarily focus on recentering and rescaling activations, Weight Standardization (WS)

targets the smoothing effects on weights. This approach results in a more favorable loss landscape and improved parameter updating. Additionally, WS guarantees batch-agnostic normalization, as it is not impacted by the batch size. The WS operation is defined as:

$$\widetilde{\boldsymbol{W}} = \frac{\boldsymbol{W} - \boldsymbol{\mu}_w}{\boldsymbol{\sigma}_w + \epsilon} \qquad (3)$$

Here, $\boldsymbol{\mu}_w$ and $\boldsymbol{\sigma}_w$ represent the mean and standard deviation of the weights calculated over all channels and dimensions, computed as:

$$\boldsymbol{\mu}_w = \frac{1}{N} \sum_{i,j} \boldsymbol{w}_{i,j}, \quad \boldsymbol{\sigma}_w = \sqrt{\frac{1}{N} \sum_{i,j} (\boldsymbol{w}_{i,j} - \boldsymbol{\mu}_w)^2} \qquad (4)$$

and $w_{i,j}$ represents the element of the weight matrix for the $i_{th}$ row and $j_{th}$ column. This normalization is applied directly to the early exit CNN layers $\phi^k$. Importantly, WS introduces no additional parameters within the CNN layers and facilitates TTA across different batch sizes. During inference, WS is computed solely from the weights, eliminating the need to accumulate batch statistics on the fly, thus achieving batch-agnostic adaptation. This allows it to be applicable to small batch sizes, thus further ensuring memory and computational efficiency during TTA and making TinyTTA suitable for deployment on MCUs. From an implementation viewpoint, for MCU deployment, we introduce a new CNN layer incorporating the WS normalization in our TinyTTA Engine (see Section 3.5) to avoid using batch normalization layers, which are not supported by most constrained hardware.

## 3.4 Training and Inference

To facilitate the learning of submodules $\theta^k$ and early-exit model parameters $\phi^k$, we introduce the concurrent training of all models on the training dataset. Given the source data, we pass them through both the pre-trained model and the submodules combined with early exit branches and compute the losses to optimize the models. The first loss, $\mathcal{L}_1$, aims to infer accurate predictions from the submodules and early-exit modules, which can be formulated as the cross-entropy (CE) for all classes. The second loss, $\mathcal{L}_2$, aims to achieve reliable self-ensembling by aligning the outputs from the submodules to be the same as the pre-trained models. The process can be formulated as:

$$\mathcal{L}_1 = \sum_{i=1}^{C} CE\left(\boldsymbol{p}_i, y\right), \quad \mathcal{L}_2 = \|\tilde{\boldsymbol{z}}_k - \boldsymbol{z}_k\|_1 \qquad (5)$$

where $\tilde{z}_k$ represents the features learned from the submodules and $\boldsymbol{z}_k$ represents the one from the pre-trained model. The final loss consists of these two losses:

$$\min_{\theta} \mathcal{L} = \lambda \mathcal{L}_1 + (1 - \lambda)\mathcal{L}_2 \qquad (6)$$

where $\lambda$ controls the trade-off between the two losses.

The training process is performed offline, without additional training costs on the device. Then, at inference time during the TTA phase, TinyTTA operates without utilizing any source data, as in the standard TTA setting [6]. The model initially calculates the entropy of the first submodule's output and compares it with a predefined entropy threshold. Based on this comparison, the exit determines whether the feature outputs should proceed to the subsequent submodules of the self-ensemble model. The process continues if the entropy is greater than the threshold; otherwise, it halts. It is worth noticing that after training, only the submodules and early exits are deployed on the device. Once deployed, only early exit branches are updated on-device, while the rest of the model remains frozen, ensuring both high TTA accuracy and low memory usage.

## 3.5 TinyTTA Engine

To enable TTA on edge devices like MCUs, we propose TinyTTA Engine as illustrated in Figure 4. The TinyTTA Engine pipeline encompasses several steps. Initially, once the model is loaded, its forward graph is constructed. During compile-time, operations are optimized through fusion, and an automatic differentiation (`autodiff`) process constructs the backward graph needed for updating. Following `autodiff`, the backward graph is optimized via a *fuse and quantize* procedure, aiming to mitigate resource limitations during execution.

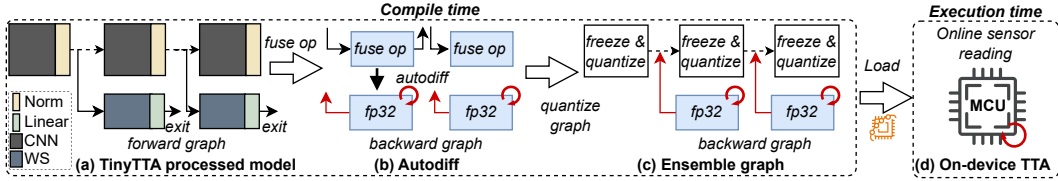

Figure 4: The TinyTTA Engine operates in two phases: compile time and execution time. During compile time, (a) given a reprocessed model, it (b) fuses backbone operations to enhance efficiency, then enables backpropagation on TinyTTA exits. Subsequently, (c) it freezes and quantizes the backbone before integration with TinyTTA exits. Finally, the model is loaded onto the MCU for (d) on-device TTA during execution time.

The cornerstone of TinyTTA Engine lies in its ability to perform backpropagation on-device. We build TinyTTA Engine upon the widely used MCU library TFLM by developing a floating-point-based backpropagation algorithm. TinyTTA Engine specifically caters to the backward computation needs of prevalent DNN operators such as `ReLU`, `FullyConnected`, `Softmax`, `Maxpool`, `Avgpool`, `Conv`, and `DepthwiseConv`. To further optimize memory efficiency, we have developed a layer-wise update strategy to reduce memory utilization alongside a dynamic memory allocation mechanism leveraging heap memory. Specifically, the interpreter-based framework of TinyTTA Engine enables memory reduction by dynamically choosing to save intermediate activations only if a layer needs updates (refer to Appendices C.2 and C.3). The TinyTTA Engine code is publicly available at `https://github.com/h-jia/TTE`.

# 4 Experimental Setup

**Distribution shifted datasets.** For comparison with baselines, our experiments were conducted using four popular corrupted datasets, including domain shift datasets CIFAR10C [23] and CIFAR100C [24], and two domain generalization datasets, namely OfficeHome [25] and PACS [26]. More details about the datasets can be found in Appendix B.

**Implementation details.** To provide a comprehensive evaluation of TinyTTA, we conducted experiments on both MPUs via a Raspberry Pi Zero 2W and MCUs using an STM32H747. Details of the hardware and MCU implementation can be found in Appendix C. The evaluation of TinyTTA on MPUs, which are equipped with relatively large memory, allows for a comparison with other state-of-the-art (SOTA) methods, thereby offering an in-depth understanding of TinyTTA's algorithmic performance. For the MPU experiments, we employed three SOTA mobile models, specifically EfficientNet [27], MobileNetV2 [28], and RegNet [29]. In particular, we utilized MobileNetV2 with a scale factor of 0.5 (MobileNetV2_x05), EfficientNet with a scaling factor of 1 (EfficientNet_b1), and RegNet with a computational cost of 200 million floating-point operations (RegNet-200m). For the MCU evaluation, we utilized the SOTA model MCUNet [5], which is the only model capable of running within the 512 KB memory constraint of MCUs. More details are discussed in Appendix D.

**Baselines.** We compared TinyTTA with several SOTA methods, including: $i$) TENT [6], which optimizes the affine parameters of batch normalization layers through entropy minimization; $ii$) CoTTA [13], which updates all model parameters using a consistency loss between student and teacher models and stochastically restores the pre-trained model; $iii$) a memory-efficient baseline EATA [7], which employs a sample selection criterion for identifying non-redundant samples and updates the model by minimizing entropy loss; $iv$) the most memory-efficient method EcoTTA [13], which freezes the original network parameters, uses AdaptBN [30] normalization layers and updates attached meta networks for memory-efficient test-time adaptation.

**Evaluation metrics.** The same evaluation metrics are utilized across all datasets, encompassing accuracy, memory usage, latency, and energy consumption. Memory usage is quantified using TinyTL [31] for the Raspberry Pi Zero 2W and with TFLite Tools[1] for MCUs, with a focus on the `tensor-arena` size. As both the MPU and MCU necessitate loading libraries for TTA, latency is

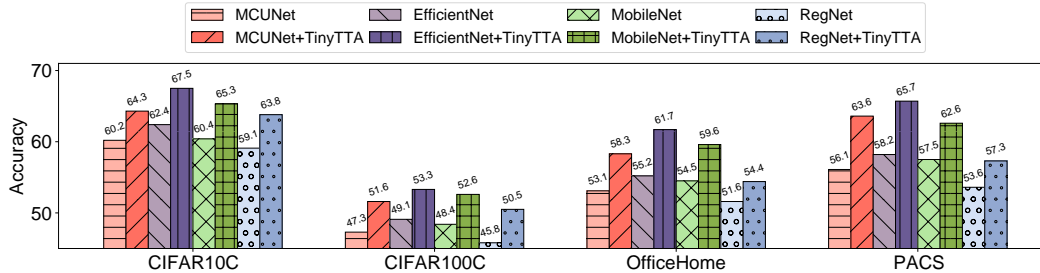

Figure 5: Comparison of model performance across four datasets, demonstrating the accuracy improvements when models are adapted using TinyTTA. Across all datasets, TinyTTA consistently boosts accuracy with respect to not adapting the model.

recorded per batch. This measurement is taken after the model stabilizes following 10 warm-up iterations of test samples in TTA. Subsequently, the latency is averaged over 100 batches.

# 5   Results

This section addresses the following questions: (1) How does TinyTTA improve performance compared to the source model without adaptation? (2) How does TinyTTA perform compared to baseline methods on MPUs? (3) How efficient is TinyTTA on typical MCUs? (4) What effect do self-ensemble, early-exits, and WS have?

## 5.1   TinyTTA vs. source models on MPUs

We first compare the performance between TinyTTA and the source model without adaptation on MPUs, as shown in Figure 5. The performance was assessed using four distinct corrupted datasets and four different model architectures. Notably, TTA is evaluated with a batch size of one, a challenging scenario for adaptation. Most existing TTA methods generally fail due to unstable statistics associated with a single batch for normalization layers. For each combination of model and dataset, TinyTTA demonstrates an average improvement of 4.3%. The most significant enhancement is observed with the EfficientNet model on the CIFAR100C dataset, where accuracy increases from 49.1% to 53.3%. Even under the constraint of adapting to a single batch of data during test time, TinyTTA consistently outperforms models without adaptation, highlighting its potential in real-world scenarios where limited data is available for adaptation.

## 5.2   TinyTTA vs. SOTA baselines on MPUs

We further compared TinyTTA with other SOTA baselines across four different datasets and four model architectures on MPUs. The results are shown in Figures 6a to 6d.

For MCUNet, as shown in Figure 6a, TinyTTA consistently outperforms other TTA methods across all datasets. Specifically, our method achieves an accuracy of 64.3%, 51.6%, 58.1%, and 63.6% on CIFAR10C, CIFAR100C, OfficeHome, and PACS datasets, respectively. In comparison, the second-best performing method, ECoTTA, only attains an accuracy of 13.1%, 5.5%, 6.2%, and 6.3% on the same datasets, significantly deteriorating the performance compared to the source model without adaptation. This is due to the small batch size that causes unstable statistics for adaptation. Notably, on CIFAR100C, OfficeHome, and PACS datasets, TinyTTA achieves substantial accuracy improvements of 46.6%, 52.1%, and 57.6% over the second-best method, respectively. In terms of memory usage, TinyTTA is exceptionally efficient, requiring 1.2× less memory than ECoTTA, 2.2× less than EATA, 2.3× less than TENT (Modulating), 4.2× less than TENT (Fine-tuning), and 6.0× less than CoTTA on average across all datasets on MPUs.

For EfficientNet, we observe a similar performance trend, as depicted in Figure 6b. In terms of accuracy, TinyTTA consistently outperforms other TTA methods across all datasets. On the CIFAR10C dataset, TinyTTA achieves an accuracy of 67.5%, which is 48.5% higher than the second-best method, CoTTA. Similarly, on CIFAR100C, OfficeHome, and PACS datasets, TinyTTA also shows 47.3%, 53.7%, and 56.7% relative improvements over the second-best method, respectively.

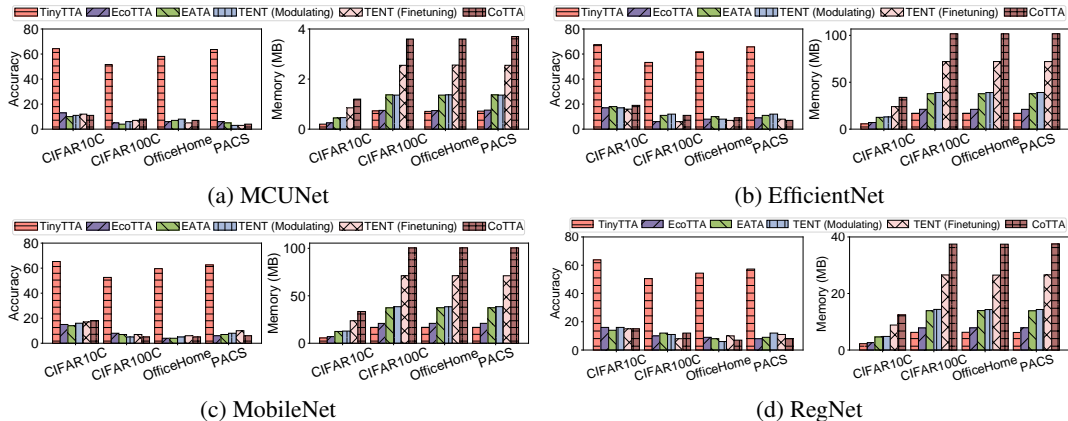

(a) MCUNet

(b) EfficientNet

(c) MobileNet

(d) RegNet

Figure 6: TTA methods performance comparison of four edge models over four datasets. (a) TinyTTA (ours) is the only method capable of performing TTA under MCU constraints while maintaining high accuracy. (b)-(d) TinyTTA (ours) achieves the best performance using minimum memory. Results are tested on severity level 5.

Regarding memory usage, our method exhibits the lowest memory footprint among all the compared methods. On the CIFAR10C dataset, TinyTTA requires 5.65 MB of memory, which is 1.2× lower than ECoTTA (7.06 MB), 2.2× lower than EATA (12.59 MB), 2.3× lower than TENT (Modulating) (13.02 MB), 4.2× lower than TENT (Fine-tuning) (24.0 MB), and 6.0× lower than CoTTA (33.88 MB). The memory usage trend remains consistent across other datasets.

For MobileNet and RegNet in Figures 6c and 6d, the results also show a similar trend as MCUNet and EfficientNet in terms of both accuracy and memory usage.

We also compared the latency and energy consumption of the different TTA methods on the Raspberry Pi Zero 2W MPU using the CIFAR10C dataset, which consists of 50,000 samples.
The results in Table 1 highlight the efficiency of TinyTTA in terms of both latency and energy consumption. Specifically, TinyTTA achieves an inference time of 0.22 seconds per sample (11,000 seconds in total) with a total energy consumption of 6.11 Wh. Compared to EATA, for example, TinyTTA reduces both latency and energy consumption by 12%. On the other hand, CoTTA shows the highest latency (312,500 seconds in total) and energy consumption (173.61 Wh), while TENT (both fine-tuning and modulating), though more efficient than CoTTA, is still outperformed by TinyTTA. Overall, TinyTTA offers the best balance between accuracy, memory, speed, and energy efficiency, making it ideal for edge AI applications.

Table 1: Latency (in seconds) and energy consumption (in Joule) per image for various methods on CIFAR10C on Raspberry Pi Zero 2W MPU.

| Method | Latency (sec) | Energy (J) |
|---|---|---|
| CoTTA | 6.25 | 12.50 |
| TENT (Finetune) | 0.51 | 1.02 |
| TENT (Modulating) | 0.51 | 1.02 |
| EATA | 0.25 | 0.50 |
| ECoTTA | 0.37 | 0.75 |
| **TinyTTA (Ours)** | 0.22 | 0.44 |

## 5.3 TinyTTA on MCUs

We now show the performance of TinyTTA on extremely resource-constrained edge devices, specifically MCUs with only 512 KB of on-chip memory—significantly smaller than those of MPUs like the Raspberry Pi Zero 2W. *Notably, TinyTTA is the only method capable of performing TTA under the STM32H747 MCU's 512 KB memory constraint while achieving superior accuracy compared to other methods.* This makes TinyTTA a promising solution for resource-constrained devices requiring efficient and accurate TTA. We examine two scenarios: with and without TinyTTA ("Inference Only") to analyze effectiveness and system overheads. The results are shown in Table 2.

In addition to the superior performance over "Inference Only", as shown in Section 5.2, TinyTTA reduces overall latency (50.7 ms vs. 55.8 ms) and energy consumption (11.5 mJ vs. 12.7 mJ) per

sample thanks to the early exit mechanism, despite requiring on-device model updates. Regarding memory usage, TinyTTA incurs small overheads of 40.2 KB for SRAM and 85.1 KB for Flash. Specifically, the additional SRAM is needed for updated parameters after adaptation, while the early exits require additional heads, taking up storage on Flash.

Table 2: MCU deployment of the baseline and TinyTTA on STM32H747 using MCUNet and CIFAR10C.

| System | Accuracy | SRAM | Flash | Latency | Energy |
|---|---|---|---|---|---|
| Inference Only | 60.2% | **82.8KB** | **290KB** | 55.8ms | 12.7mJ |
| TinyTTA (update) | **64.3%** | 123KB | 375KB | **50.7ms** | **11.5mJ** |

*Notably, our TinyTTA Engine enables fast (low latency) and efficient (low energy) on-device adaptation with only a few tens of KBs of extra memory on extremely resource-limited IoT devices.*

## 5.4 Ablation Study

Figure 7 illustrates the trade-offs between accuracy and memory usage (in MB) across different configurations of our method for four datasets using MCUNet. The configuration represented by ★ (TinyTTA using all components: self-ensemble, early-exits, and WS) achieves the best balance, with the highest accuracy and optimal memory usage. In contrast, ■ (TinyTTA without WS) shows a drop in accuracy by approximately 12-15% and no change in memory usage. The ▲ (TinyTTA without early exits and WS) configuration reduces accuracy by about 44.5-54.5% and increases memory usage by 5×, highlighting the critical role of early exits. Finally, ♦ (without all components) exhibits the lowest accuracy, decreasing by around 40.3-50.3%, and the highest memory footprint increased by 5-6×, underscoring the necessity of integrating all components. The ablation study clearly demonstrates that our method, when leveraging all components, achieves the optimal trade-off for practical applications.

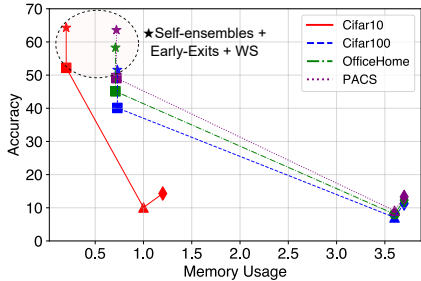

Figure 7: Ablation study. ★ represents TinyTTA using all components. ■ represents without WS. ▲ represents without early-exits and WS, and ♦ means without all components.

## 6 Conclusion

This paper presented TinyTTA, which, for the first time, enables TTA on resource-limited edge devices such as MCUs. By introducing a self-ensemble framework, an early-exit policy, and weight standardization, our method overcomes the limitations of traditional TTA methods, enabling efficient yet competitive on-device TTA on edge devices. We also offer a new library, TinyTTA Engine, to unlock the development of TTA for various applications on MCUs.

## Limitations and Future Work

TinyTTA has only been tested with a batch size of one in the setting to be deployed on a single type of MCU with image data. Additionally, TinyTTA requires retraining the model via self-ensemble. However, the underlying TinyTTA Engine offers a robust foundation for extending its capabilities. We anticipate that with further development, our method and the TinyTTA Engine library could be adapted for a wider array of applications, encompassing various types of MCUs and diverse data modalities beyond just images. Future research will involve investigating the potential of TinyTTA to process different types of data, such as video and inertial measurement unit (IMU) data, thereby broadening its usability across different real-world applications.

## Broader Impacts

Deploying TTA on microcontrollers is particularly useful for applications that need quick decision-making and operate in resource-limited, changing environments. Unlike traditional model inference,

where the model's weights stay the same, TTA requires the model to update its weights based on shifts in input data distribution. This makes it challenging to maintain accuracy while being deployable on MCUs. Our TinyTTA framework addresses this issue, enabling TTA on MCUs for the first time.

## 7    Acknowledgment

This work is supported by ERC through Project 833296 (EAR), and Nokia Bell Labs through a donation. We also acknowledge financial support from the PNRR MUR project PE0000013-FAIR (Italy) - CUP H23C22000860006.

## Footnotes

[1]https://github.com/eliberis/tflite-tools

[2]https://github.com/ARMmbed/mbed-os/tree/master

[3]https://github.com/tensorflow/tflite-micro/tree/main/tensorflow/lite/micro

[4]https://github.com/eliberis/tflite-tools

[5]https://github.com/chenyaofo/pytorch-cifar-models

[6]https://huggingface.co/google/efficientnet-b1

[7]https://github.com/huggingface/pytorch-image-models

## References

[1] Muyang Lin, Ziyang Zhang, Xiaoxiang Gao, Yizhou Bian, Ray S Wu, Geonho Park, Zhiyuan Lou, Zhuorui Zhang, Xiangchen Xu, Xiangjun Chen, et al. A fully integrated wearable ultrasound system to monitor deep tissues in moving subjects. *Nature Biotechnology*, 42(3):448–457, 2024.

[2] Artúr István Károly, Péter Galambos, József Kuti, and Imre J Rudas. Deep learning in robotics: Survey on model structures and training strategies. *IEEE Transactions on Systems, Man, and Cybernetics: Systems*, 51(1):266–279, 2020.

[3] Shuaicheng Niu, Jiaxiang Wu, Yifan Zhang, Zhiquan Wen, Yaofo Chen, Peilin Zhao, and Mingkui Tan. Towards stable test-time adaptation in dynamic wild world. In *The Eleventh International Conference on Learning Representations*, 2023.

[4] Francesco Croce, Maksym Andriushchenko, Vikash Sehwag, Edoardo Debenedetti, Nicolas Flammarion, Mung Chiang, Prateek Mittal, and Matthias Hein. Robustbench: a standardized adversarial robustness benchmark. In *Proceedings of the Neural Information Processing Systems Track on Datasets and Benchmarks*, 2021.

[5] Ji Lin, Ligeng Zhu, Wei-Ming Chen, Wei-Chen Wang, Chuang Gan, and Song Han. On-device training under 256kb memory. *Advances in Neural Information Processing Systems*, 35:22941–22954, 2022.

[6] Dequan Wang, Evan Shelhamer, Shaoteng Liu, Bruno A. Olshausen, and Trevor Darrell. Tent: Fully test-time adaptation by entropy minimization. In *9th International Conference on Learning Representations*, 2021.

[7] Shuaicheng Niu, Jiaxiang Wu, Yifan Zhang, Yaofo Chen, Shijian Zheng, Peilin Zhao, and Mingkui Tan. Efficient test-time model adaptation without forgetting. In *International conference on machine learning*, pages 16888–16905. PMLR, 2022.

[8] Skyler Seto, Barry-John Theobald, Federico Danieli, Navdeep Jaitly, and Dan Busbridge. Realm: Robust entropy adaptive loss minimization for improved single-sample test-time adaptation. In *Proceedings of the IEEE/CVF Winter Conference on Applications of Computer Vision*, pages 2062–2071, 2024.

[9] Ji Lin, Wei-Ming Chen, Yujun Lin, john cohn, Chuang Gan, and Song Han. Mcunet: Tiny deep learning on iot devices. In H. Larochelle, M. Ranzato, R. Hadsell, M.F. Balcan, and H. Lin, editors, *Advances in Neural Information Processing Systems*, volume 33, pages 11711–11722. Curran Associates, Inc., 2020.

[10] Benoit Jacob, Skirmantas Kligys, Bo Chen, Menglong Zhu, Matthew Tang, Andrew Howard, Hartwig Adam, and Dmitry Kalenichenko. Quantization and Training of Neural Networks for Efficient Integer-Arithmetic-Only Inference. In *IEEE Conference on Computer Vision and Pattern Recognition (CVPR)*, 2018.

[11] Robert David, Jared Duke, Advait Jain, Vijay Janapa Reddi, Nat Jeffries, Jian Li, Nick Kreeger, Ian Nappier, Meghna Natraj, Tiezhen Wang, et al. Tensorflow lite micro: Embedded machine learning for tinyml systems. *Proceedings of Machine Learning and Systems*, 3:800–811, 2021.

[12] Siyuan Qiao, Huiyu Wang, Chenxi Liu, Wei Shen, and Alan Yuille. Micro-batch training with batch-channel normalization and weight standardization. *arXiv preprint arXiv:1903.10520*, 2019.

[13] Junha Song, Jungsoo Lee, In So Kweon, and Sungha Choi. Ecotta: Memory-efficient continual test-time adaptation via self-distilled regularization. In *Proceedings of the IEEE/CVF Conference on Computer Vision and Pattern Recognition*, pages 11920–11929, 2023.

[14] Yu Sun, Xiaolong Wang, Zhuang Liu, John Miller, Alexei Efros, and Moritz Hardt. Test-time training with self-supervision for generalization under distribution shifts. In *International conference on machine learning*, pages 9229–9248. PMLR, 2020.

[15] Hyesu Lim, Byeonggeun Kim, Jaegul Choo, and Sungha Choi. TTN: A domain-shift aware batch normalization in test-time adaptation. In *The Eleventh International Conference on Learning Representations*, 2023.

[16] Taesik Gong, Jongheon Jeong, Taewon Kim, Yewon Kim, Jinwoo Shin, and Sung-Ju Lee. Note: Robust continual test-time adaptation against temporal correlation. *Advances in Neural Information Processing Systems*, 35:27253–27266, 2022.

[17] Taesik Gong, Yewon Kim, Taeckyung Lee, Sorn Chottananurak, and Sung-Ju Lee. Sotta: Robust test-time adaptation on noisy data streams. *Advances in Neural Information Processing Systems*, 36, 2024.

[18] Junyuan Hong, Lingjuan Lyu, Jiayu Zhou, and Michael Spranger. Mecta: Memory-economic continual test-time model adaptation. In *The Eleventh International Conference on Learning Representations*, 2022.

[19] Huaxiu Yao, Caroline Choi, Bochuan Cao, Yoonho Lee, Pang Wei W Koh, and Chelsea Finn. Wild-time: A benchmark of in-the-wild distribution shift over time. *Advances in Neural Information Processing Systems*, 35:10309–10324, 2022.

[20] Bowen Zhao, Chen Chen, and Shu-Tao Xia. Delta: Degradation-free fully test-time adaptation. In *The Eleventh International Conference on Learning Representations*, 2022.

[21] Surat Teerapittayanon, Bradley McDanel, and H. T. Kung. Branchynet: Fast inference via early exiting from deep neural networks. In *23rd International Conference on Pattern Recognition, ICPR 2016*, pages 2464–2469. IEEE, 2016.

[22] Yigitcan Kaya, Sanghyun Hong, and Tudor Dumitras. Shallow-deep networks: Understanding and mitigating network overthinking. In Kamalika Chaudhuri and Ruslan Salakhutdinov, editors, *Proceedings of the 36th International Conference on Machine Learning, ICML 2019*, volume 97 of *Proceedings of Machine Learning Research*, pages 3301–3310. PMLR, 2019.

[23] Alex Krizhevsky, Geoffrey Hinton, et al. Learning multiple layers of features from tiny images. 2009.

[24] Collin Burns and Jacob Steinhardt. Limitations of post-hoc feature alignment for robustness. In *Proceedings of the IEEE/CVF Conference on Computer Vision and Pattern Recognition*, pages 2525–2533, 2021.

[25] Hemanth Venkateswara, Jose Eusebio, Shayok Chakraborty, and Sethuraman Panchanathan. Deep hashing network for unsupervised domain adaptation. In *Proceedings of the IEEE conference on computer vision and pattern recognition*, pages 5018–5027, 2017.

[26] Da Li, Yongxin Yang, Yi-Zhe Song, and Timothy M Hospedales. Deeper, broader and artier domain generalization. In *Proceedings of the IEEE international conference on computer vision*, pages 5542–5550, 2017.

[27] Mingxing Tan and Quoc V. Le. Efficientnet: Rethinking model scaling for convolutional neural networks. In Kamalika Chaudhuri and Ruslan Salakhutdinov, editors, *Proceedings of the 36th International Conference on Machine Learning, ICML 2019, 9-15 June 2019, Long Beach, California, USA*, volume 97 of *Proceedings of Machine Learning Research*, pages 6105–6114. PMLR, 2019.

[28] Mark Sandler, Andrew Howard, Menglong Zhu, Andrey Zhmoginov, and Liang-Chieh Chen. Mobilenetv2: Inverted residuals and linear bottlenecks. In *Proceedings of the IEEE conference on computer vision and pattern recognition*, pages 4510–4520, 2018.

[29] Ilija Radosavovic, Raj Prateek Kosaraju, Ross Girshick, Kaiming He, and Piotr Dollár. Designing network design spaces. In *Proceedings of the IEEE/CVF conference on computer vision and pattern recognition*, pages 10428–10436, 2020.

[30] Steffen Schneider, Evgenia Rusak, Luisa Eck, Oliver Bringmann, Wieland Brendel, and Matthias Bethge. Improving robustness against common corruptions by covariate shift adaptation. *Advances in neural information processing systems*, 33:11539–11551, 2020.

[31] Han Cai, Chuang Gan, Ligeng Zhu, and Song Han. Tinytl: Reduce memory, not parameters for efficient on-device learning. *Advances in Neural Information Processing Systems*, 33:11285–11297, 2020.

[32] Colby Banbury, Chuteng Zhou, Igor Fedorov, Ramon Matas, Urmish Thakker, Dibakar Gope, Vijay Janapa Reddi, Matthew Mattina, and Paul Whatmough. Micronets: Neural network architectures for deploying tinyml applications on commodity microcontrollers. *Proceedings of machine learning and systems*, 3:517–532, 2021.

[33] Viet Anh Trinh, Hassan Salami Kavaki, and Michael I. Mandel. Importantaug: A data augmentation agent for speech. In *IEEE International Conference on Acoustics, Speech and Signal Processing, ICASSP 2022, Virtual and Singapore, 23-27 May 2022*, pages 8592–8596. IEEE, 2022.

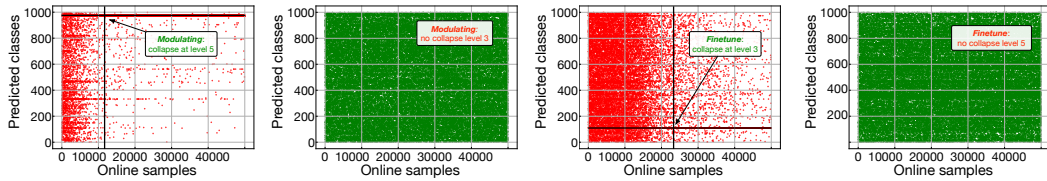

(a) Modulating at level 5    (b) Modulating at level 3    (c) Fine-tuning at level 3    (d) Fine-tuning at level 5

Figure 8: Comparison between modulating and fine-tuning using online test-time entropy minimization. (a-b) For larger domain shifts (e.g., severity level 5), the modulating model is more prone to collapse and predict fewer classes. (c-d) For smaller domain shifts (e.g., severity level 3), modulating is more robust than fine-tuning. All experiments are conducted on ImageNet-C for fog noise and ResNet50. Refer to Section B for more details about severity levels.

# A    Appendix/supplemental material

**Entropy Minimization**. In test-time adaptation, the objective is to fine-tune the model parameters during the testing phase to better align with the test data distribution without the labels. One technique used for this purpose is entropy minimization [6], which aims to make the model's predictions more confident on the test data. For a classification model, the entropy of the model's prediction for a data point is calculated using the formula:

$$H(\hat{p}) = -\sum_{c=1}^{C} \hat{p}_c \log(\hat{p}_c) \tag{7}$$

where $H(\hat{p})$ represents entropy, $C$ is the number of classes, and $\hat{p}_c$ is the predicted probability of the data point belonging to class $c$.

**Modulating and Finetune TTA.** Our empirical analyses in Figures 8d and 8c demonstrate the performance of fine-tuning TTA for samples with severity levels 3 and 5, respectively. In contrast, Figures 8a and 8b illustrate the outcomes of modulation-based methods for the same severity levels. For fine-tuning TTA, while it can adapt to severe data corruption (e.g., severity level 5) by tuning the entire model, it struggles with milder domain shifts (e.g., severity level 3), leading to a model collapse where it predicts only a limited range of classes. This suggests that fine-tuning is more effective for significant domain shifts but becomes prone to collapse under minor shifts, resulting in limited class predictions. On the other hand, modulation-based TTA methods exhibit a different pattern. With milder severity (e.g., level 3) as shown in Figure 8b, the modulation-based method accurately predicts samples across various classes, indicated by the dispersed points representing the 1000 classes. However, under severe data corruption (e.g., level 5), modulation-based methods tend to predict only a limited number of classes with sequential test samples, as illustrated in Figure 8a by the black line showing overlapping predicted classes for a single class. This suggests a paucity in existing methods, indicating that different mechanisms might be required to address varying levels of distribution shifts within a universal framework, which TinyTTA achieves.

# B    Datasets

CIFAR10C and CIFAR100C feature 15 types of corruption, categorized into four main groups: noise, blur, weather, and digital, each with five severity levels. The OfficeHome dataset comprises four domains $d \in \{art, clipart, product, real\}$, containing 15,588 examples with dimensions (3, 224, 224) and 65 classes. Similarly, the PACS dataset comprises four domains $d \in \{art, cartoons, photos, sketches\}$, containing 9,991 examples with dimensions (3, 224, 224) and 7 classes.

## C   Implementation

We now introduce the hardware and software implementation of TinyTTA. Following this, we will discuss one of the core tasks of TinyTTA, which is the MCU implementation.

### C.1   Hardware

The self-ensemble training phase of our framework was conducted on a Linux server equipped with an Intel(R) Xeon(R) Gold 5218 CPU @ 2.30GHz and an NVIDIA Quadro RTX 6000 GPU, where we pre-trained the shared backbone and multiple heads. For the deployment phase, the system components were implemented on two types of edge devices. The first MPU was a Raspberry Pi Zero 2W featuring an ARM Cortex-A53 processor with distinct memory specifications. The second MCU we used is the STM32H747XI, which possesses a dual-core (ARM Cortex M4 and M7) architecture. We only use the ARM Cortex M7 core, reflecting a typical single-core MCU design. This choice constrained the SRAM and embedded Flash capacities to 512 KB and 1 MB for STM32H747XI, respectively. These two platforms are representative of a broad spectrum of devices commonly used in edge computing. Specifically, as the MCU is based on the ARM Cortex-M architecture, TinyTTA can be seamlessly deployed across a range of similar ARM Cortex-M-based MCUs, demonstrating the versatility of our approach in resource-constrained environments.

### C.2   Software

The pre-training stage of TinyTTA, executed on a standard Linux server, utilized PyTorch version 1.10.2. For the meta-learner module, Python, NumPy, and TensorFlow Lite Micro (TFLM) were employed due to their compatibility and efficient integration. TFLM was specifically selected for its effectiveness and flexibility in deployment across diverse hardware platforms and pre-trained models without re-generating the code that embeds architectures, weights, and execution graphs and logic. The efficacy of TinyTTA was evaluated by analyzing the required system resources to enable TTA regarding memory usage of SRAM and Flash, end-to-end latency, and energy consumption. The process of transitioning a PyTorch model to these MCUs with TensorFlow Lite involved a multi-stage conversion: initially to ONNX format, then to TensorFlow, and finally to TensorFlow Lite, utilizing appropriate converters at each step. The execution of the TensorFlow Lite model on the MCUs was facilitated by TFLM and Mbed OS.

TensorFlow Lite for Microcontrollers (TFLM) [11] offers a platform tailored for executing machine learning models on tiny devices, circumventing the necessity for conventional operating systems or standard C/C++ libraries. TFLM's interpreter-based methodology has its advantages, providing adaptability to a wide set of edge devices and pre-trained models with minimal runtime overhead for deep neural network computations. However, TFLM's principal shortcoming lies in its incapacity for on-device training, confining models to a static state post-deployment for solely inference purposes. Recognizing the advantages of TFLM's minimal runtime overhead and high level of flexibility, we developed our TinyTTA Engine framework by augmenting a new functionality, on-device adaptation during runtime, to TFLM.

### C.3   MCU Implementation

We deployed TinyTTA on the STM32H747XI. The deployment process involves loading the necessary libraries from Mbed OS[2] and acquiring the GCC ARM toolchain and the `mbed-cli` dev tool. For model management, the TFLM[3] framework is set up in the project environment, alongside `mbed-os` components. TFLite Tools[4] are employed to approximate rough SRAM requirements. The model is then converted into a `C` array format suitable for MCU deployment.

In addition, we enabled on-device adaptation of TinyTTA Engine by implementing a backward pass computation and extending this functionality on top of the floating-point-based operators (mentioned in Section 3.5) that previously supported only forward-pass computation in the TFLM framework. In detail, we implemented the gradient calculation regarding weights, biases, and activations in C for

each operator. Furthermore, we developed a layer-wise update logic that allows TinyTTA Engine to select which layer to update during on-device TTA to further reduce the memory overhead on SRAM and the required computational costs, as gradient calculation for weights can be omitted for non-updated layers.

Lastly, the increased binary size derived from our backpropagation implementation in TinyTTA is merely 8.6 KB as it is implemented on top of the existing operator codebase and third-party libraries. This shows that TinyTTA enables on-device TTA on MCUs for the first time, with minimal storage overhead.

### C.4 TinyTTA vs. TinyEngine

As our TinyTTA Engine enables on-device training for TTA, in this section we discuss the state-of-the-art on-device training framework for MCUs, known as TinyEngine [5]. Specifically, TinyEngine focuses on on-device training (with labeled data), employing automatic differentiation operations at compile-time and utilizing code generation to minimize runtime overhead. The framework statically pre-determines the layers and channels to be updated before deployment, allowing these updates to be executed at runtime. While this approach is efficient, it imposes constraints on post-deployment adaptability, as any model update requires recompilation for the target device. Furthermore, due to the code generation process, the model's architecture, weights, and layer dimensions are hard-coded into the binary file, which necessitates replacing the entire executable to modify the model.

In contrast, our TinyTTA Engine focuses on test-time adaptation with unlabeled data. It enables dynamic adaptation during inference by allowing early exits at submodules, specifically for high-entropy samples, thereby enabling reliable TTA. To adapt TinyEngine for TTA, the only practical solution is to use TENT [6], which fine-tunes the model via entropy minimization. In our evaluation, we compared TinyEngine using TENT on a Raspberry Pi Zero 2W (with batch size 1) to our TinyTTA, measuring performance in terms of accuracy. The experimental setup is the same as described in Appendices B and C, and the results are summarized in Figure 9.

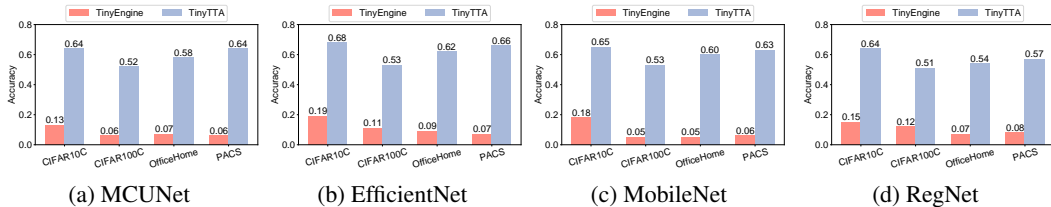

| (a) MCUNet | (b) EfficientNet | (c) MobileNet | (d) RegNet |

Figure 9: Comparison of accuracy between TinyEngine (using TENT) and TinyTTA (ours) on different datasets and model architectures.

The results demonstrate that TinyTTA, powered by the TinyTTA Engine, significantly outperforms TinyEngine across all datasets and model architectures due to its ability to dynamically exit high-entropy samples without the need for recompilation, making it a more practical solution for real-world deployment, where data distributions may evolve over time.

## D   Model size

Compared with ResNet50, on which the majority of TTA works focused, models targeting mobile devices are much smaller in terms of model parameters. Specifically, MobileNetV2_×05[5], Efficient-Net_b1[6], and RegNet-200m [7] have parameters totaling 23.71 MB, 24 MB, and 8.85 MB, respectively. In comparison, ResNet50 is approximately 40 times larger than MCUNet in terms of the number of parameters, which only has 2.6 MB. The MCUNet model, optimized for 256KB SRAM and 1MB Flash, was pre-trained on ImageNet and fine-tuned on CIFAR-10 and CIFAR-100 datasets for 50

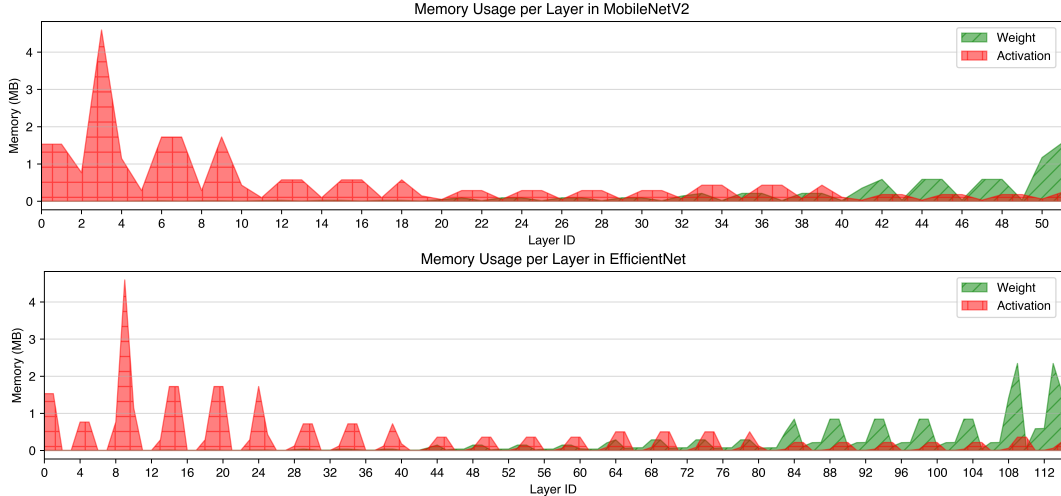

Figure 10: Memory usage of different layers for MobileNetV2 and EfficientNet.

epochs, following the methodology outlined in [31]. To further enhance the model's adaptability, we fine-tuned MCUNet for an additional 15 layers, enabling one-batch online adaptation.

As shown in Figure 10, similar to ResNet50 (cf. Figure 3), edge models such as MobileNetV2 and EfficientNet exhibit similar layerwise memory usage patterns, where adjacent layers show similar memory usage. For example, in MobileNetV2, we can group layers with similar memory usage into submodules (i.e., layers 1-11 for submodule 1, 12-20 for submodule 2, 21-40 for submodule 3, and 41-52 for submodule 4) for subsequent early exits. Similarly, for EfficientNet, we can group layers 1-26, 27-42, 43-81, and 82-115 for each submodule. Other per-layer model memory profiling can be accessed via related works, such as MCUNetV3 [5], where layers are grouped as 1-11, 12-18, 19-32, and 33-42. In RegNet, the layers are grouped as 1-6, 7-10, 11-23, and 24-42.

Regarding the computational overhead introduced by early-exit branches, we analyzed the increase in model size and the number of parameters for the early exits of three different models across various datasets. Note that the last exit layer retains the same memory and parameters as the original model. Table 3 summarizes these results.

Table 3: Memory overhead and number of parameters for early exits across models and datasets.

| Model | Dataset | 1st Exit | | 2nd Exit | | 3rd Exit | |
|---|---|---|---|---|---|---|---|
| | | Mem (MB) | Params (KB) | Mem (MB) | Params (KB) | Mem (MB) | Params (KB) |
| MobileNet | CIFAR10C | 0.01 | 2.70 | 0.03 | 8.46 | 0.07 | 17.29 |
| | CIFAR100C | 0.04 | 11.43 | 0.10 | 25.83 | 0.17 | 43.30 |
| | OfficeHome | 0.03 | 8.03 | 0.07 | 19.07 | 0.13 | 33.19 |
| | PACS | 0.01 | 2.41 | 0.03 | 7.88 | 0.06 | 16.42 |
| EfficientNet | CIFAR10C | 0.02 | 5.19 | 0.17 | 44.17 | 0.52 | 137.10 |
| | CIFAR100C | 0.07 | 18.24 | 0.33 | 87.46 | 0.75 | 197.67 |
| | OfficeHome | 0.05 | 13.17 | 0.27 | 70.62 | 0.66 | 174.11 |
| | PACS | 0.02 | 4.76 | 0.16 | 42.73 | 0.52 | 135.08 |
| RegNet | CIFAR10C | 0.10 | 25.09 | 0.53 | 140.22 | 0.53 | 140.22 |
| | CIFAR100C | 0.15 | 38.86 | 0.66 | 173.43 | 0.66 | 173.43 |
| | OfficeHome | 0.13 | 33.51 | 0.61 | 160.51 | 0.61 | 160.51 |
| | PACS | 0.09 | 24.63 | 0.53 | 139.11 | 0.53 | 139.11 |

Results show that the addition of early-exit branches results in relatively low memory overhead and parameter increases for all models. Specifically, $i$) for MobileNet, the overhead ranges from 0.01 MB to 0.17 MB across different datasets, which is negligible compared to typical memory sizes (e.g., 512 MB or larger memory configurations); $ii$) EfficientNet exhibits a slightly higher memory and parameter increase due to its more complex architecture, but still within acceptable limits for most resource-constrained devices, ranging from 0.02 MB to 0.75 MB depending on the dataset and exit

point; $iii$) In comparison, RegNet has a memory overhead ranging from 0.10 MB to 0.66 MB, which still remains low relative to the overall system memory available on most edge devices.

## E    Comparison of WS and GN

Figure 11 presents the performance comparison of TinyTTA using different normalization layers, specifically Weight Standardization (WS) and Group Normalization (GN) [3, 8], across varying levels of distribution shifts in the CIFAR-10C dataset (L1 to L5). The results are averaged over three separate runs with different random seeds. TinyTTA with WS consistently outperforms TinyTTA with GN on all datasets. For instance, on dataset L1, TinyTTA+WS achieves an accuracy of 83.2 compared to 73.4 for TinyTTA+GN. This performance gap persists across the other datasets, with TinyTTA+WS maintaining higher accuracy levels (78.3, 73.6, 68.7, and 64.3) compared to TinyTTA+GN (67.2, 63.1, 57.9, and 53.2) on L2,

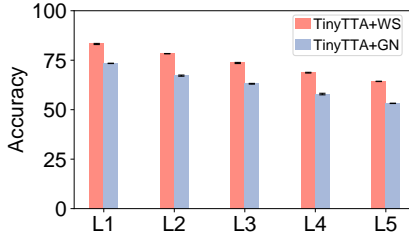

Figure 11: Ablation study at all levels (L) of distribution shift. Tests are performed on CIFAR10C.

L3, L4, and L5, respectively. The observed differences suggest that WS is more effective than GN in maintaining model accuracy under distribution shifts, likely due to its ability to better standardize the weights during the adaptation process, thus enhancing the robustness and generalization capability of TinyTTA on resource-constrained MCUs.

## F    Impact of Changing Distribution Shifts

When examining the results across different datasets for TinyTTA, as shown in Figure 11, it is evident that the severity of distribution shifts significantly impacts the model's performance. As the datasets progress from L1 to L5, there is a noticeable decline in accuracy, indicating increasing levels of difficulty in adapting to the distribution shifts. Specifically, TinyTTA achieves an accuracy of 83.2 on L1, but this performance drops to 78.3 on L2, 73.6 on L3, 68.7 on L4, and finally 64.3 on L5. This downward trend suggests that as the severity of the shifts increases, the model finds it progressively harder to maintain high accuracy. Despite this challenge, TinyTTA demonstrates a relatively strong ability to adapt, maintaining a higher level of accuracy compared to methods using other normalization strategies. This highlights the robustness of WS in handling varying degrees of distribution shifts, making it a suitable choice for deployment in resource-constrained microcontroller environments where consistent performance is crucial.

## G    Choice of Entropy Thresholds

The entropy thresholds represent the level of distribution shift in data during adaptation made by each model. A higher entropy threshold indicates a lower level of confidence in the model's predictions, while a lower entropy threshold suggests more confident predictions. The grid search results indicate the optimal entropy thresholds for four models—MCUNet, EfficientNet, MobileNet, and RegNet—across four datasets: CIFAR10C, CIFAR100C, OfficeHome, and PACS. For the CIFAR10C dataset, which comprises 10 classes, the entropy thresholds are 1.12, 1.34, 1.53, and 1.26 for MCUNet, EfficientNet, MobileNet, and RegNet, respectively. For CIFAR100C, containing 100 classes, the thresholds are 3.26, 2.58, 3.1, and 2.31. For the OfficeHome dataset, which has 65 classes, the thresholds are 2.53, 2.16, 2.28, and 2.44. For PACS, the thresholds are 1.58, 1.43, 1.14, and 1.33. These values suggest that entropy thresholds vary significantly depending on the model and the dataset, reflecting the need for model-specific and dataset-specific tuning to achieve optimal performance.

## H    Experimental Results on Distribution-Shifted Audio Data

In this section, we present the results of our experimental evaluation on distribution-shifted audio data, aiming to demonstrate the adaptive capabilities of TinyTTA with data modalities beyond

images. Specifically, we utilized a pre-trained MicroNets model [32], which was originally trained on the Speech Commands V2 dataset, achieving an accuracy of 86%. This dataset consists of 35 keywords, including "yes," "no," and "forward," among others. To evaluate the robustness of the model in real-world scenarios, we tested it on the Musan Keywords Spotting test dataset [33], which includes the same 35 speech commands but under various real-world noise conditions such as dial tones, fax machine noises, car idling, thunder, wind, footsteps, rain, and animal noises.

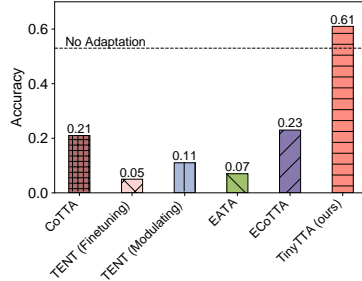

Figure 12: Performance comparison of MicroNets model on the Musan Keywords Spotting test dataset using different TTA methods.

The hyperparameters setting for TinyTTA is as follows: a learning rate of $1 \times 10^{-5}$, a batch size of 1, the SGD optimizer with a momentum of 0.9, and a self-ensemble of early exit layers at depths [3, 5, 7]. Figure 12 shows the comparative results of different TTA methods. Specifically, the pre-trained model experienced a significant performance drop, losing approximately 33% accuracy when applied to the distribution-shifted Musan dataset without any adaptation. This highlights the challenge of adapting models to real-world noisy environments. Instead, TinyTTA achieved an 8% improvement in accuracy compared to no adaptation, demonstrating strong resilience to various types of environmental noise. Finally, TinyTTA outperformed all other tested methods, achieving the highest accuracy of 0.61. Notably, the second-best method, CoTTA, achieved only 0.23 accuracy, underscoring the effectiveness of TinyTTA in maintaining performance under substantial distribution shifts, also with distribution-shifted audio data.

## I Comparison with Updating Bias Only

In this section, we explore the effects of fine-tuning only the bias, as implemented in TinyTL [31], applied to the TTA setting. We implemented a bias-only fine-tuning strategy via entropy minimization and compared it against adjusting the exits, as proposed in our TinyTTA. Specifically, during TTA, only the model's biases are updated, with no retraining of other parameters. Results in Figure 13 show that fine-tuning only the bias is insufficient for robust TTA performance, whereas integrating early exit branches and updating them in our TinyTTA demonstrates stable performance in scenarios involving distribution shifts across different datasets and models.

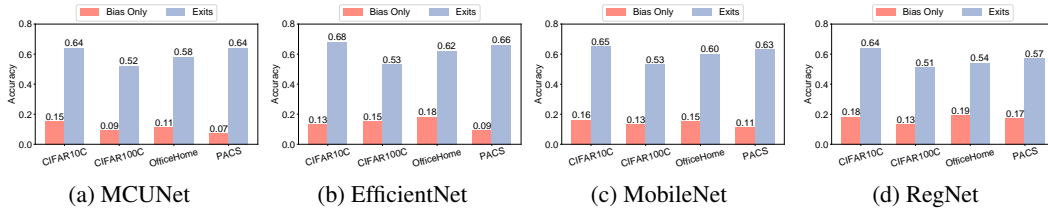

Figure 13: Comparison of bias-only fine-tuning vs. exits adjustment across different models and datasets.

## J Comparison with non-TTA methods

To better provide an understanding of the advantages of implementing TTA in resource-constrained environments, we present the comparison of TinyTTA with non-TTA methods. In particular, we implemented a new baseline using Test-Time Training [14] (TTT) with self-supervised rotation classification. We used SGD with momentum (0.9), a learning rate of $1e - 5$, an augmentation size of 20, a batch size of 1, and $\lambda = 0.9$. Tests are performed on the Raspberry Pi Zero 2W.

The experimental results, as shown in Figure 14, demonstrate a significant improvement in accuracy across all models and datasets when using TinyTTA compared to applying Test-Time Training. For example, on the CIFAR10C dataset, TinyTTA boosts MCUNet's accuracy from 16% to 64% and EfficientNet's from 18% to 68%. This trend is consistent across all datasets, with TinyTTA providing

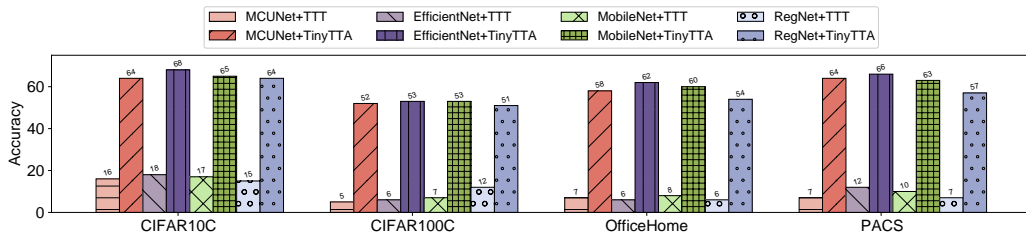

Figure 14: Comparison of Test-Time Training (TTT) and TinyTTA in terms of accuracy on various models with a batch size of one. TinyTTA significantly improves accuracy across all models and datasets.

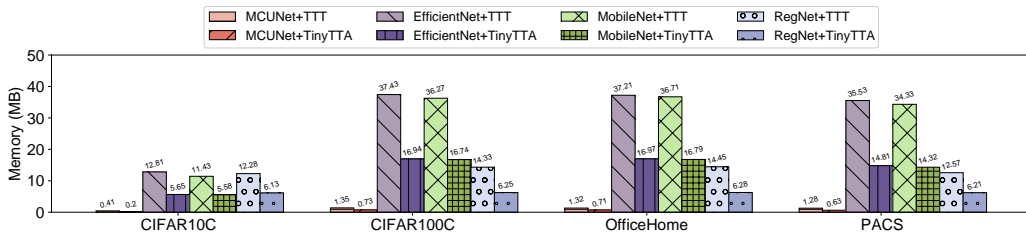

Figure 15: Comparison of Test-Time Training (TTT) and TinyTTA in terms of memory usage (in MB) on various models with a batch size of one. TinyTTA significantly improves accuracy across all models and datasets.

accuracy gains between 40-50% for all models. Interestingly, TinyTTA also reduces memory usage compared to TTT on the Raspberry Pi Zero 2W across all models and datasets, as shown in Figure 15, highlighting TinyTTA's efficiency not only in improving accuracy but also in minimizing resource consumption, making it particularly suitable for resource-constrained environments.

